# Accuracy at the Top

**Stephen Boyd**
Stanford University
Packard 264
Stanford, CA 94305
boyd@stanford.edu

**Corinna Cortes**
Google Research
76 Ninth Avenue
New York, NY 10011
corinna@google.com

**Mehryar Mohri**
Courant Institute and Google
251 Mercer Street
New York, NY 10012
mohri@cims.nyu.edu

**Ana Radovanovic**
Google Research
76 Ninth Avenue
New York, NY 10011
anaradovanovic@google.com

## Abstract

We introduce a new notion of classification accuracy based on the top $\tau$-quantile values of a scoring function, a relevant criterion in a number of problems arising for search engines. We define an algorithm optimizing a convex surrogate of the corresponding loss, and discuss its solution in terms of a set of convex optimization problems. We also present margin-based guarantees for this algorithm based on the top $\tau$-quantile value of the scores of the functions in the hypothesis set. Finally, we report the results of several experiments in the bipartite setting evaluating the performance of our solution and comparing the results to several other algorithms seeking high precision at the top. In most examples, our solution achieves a better performance in precision at the top.

## 1 Introduction

The accuracy of the items placed near the top is crucial for many information retrieval systems such as search engines or recommendation systems, since most users of these systems browse or consider only the first $k$ items. Different criteria have been introduced in the past to measure this quality, including the precision at $k$ (Precision@$k$), the normalized discounted cumulative gain (NDCG) and other variants of DCG, or the mean reciprocal rank (MRR) when the rank of the most relevant document is critical. A somewhat different but also related criterion adopted by [1] is based on the position of the top irrelevant item.

Several machine learning algorithms have been recently designed to optimize these criteria and other related ones [6, 12, 11, 21, 7, 14, 13]. A general algorithm inspired by the structured prediction technique SVMStruct [22] was incorporated in an algorithm by [15] which can be used to optimize a convex upper bound on the number of errors among the top $k$ items. The algorithm seeks to solve a convex problem with exponentially many constraints via several rounds of optimization with a smaller number of constraints, augmenting the set of constraints at each round with the most violating one. Another algorithm, also based on structured prediction ideas, is proposed in an unpublished manuscript of [19] and covers several criteria, including Precision@$k$ and NDCG. A regression-based solution is suggested by [10] for DCG in the case of large sample sizes. Some other methods have also been proposed to optimize a smooth version of a non-convex cost function in this context [8]. [1] discusses an optimization solution for an algorithm seeking to minimize the position of the top irrelevant item.

However, one obvious shortcoming of all these algorithms is that the notion of top $k$ does not generalize to new data. For what $k$ should one train if the test data in some instances is half the size and in other cases twice the size? In fact, no generalization guarantee is available for such precision@$k$ optimization or algorithm.

A more principled approach in all the applications already mentioned consists of designing algorithms that optimize accuracy in some top fraction of the scores returned by a real-valued hypothesis. This paper deals precisely with this problem. The desired objective is to learn a scoring function that is as accurate as possible for the items whose scores are above the top $\tau$-quantile. To be more specific, when applied to a set of size $n$, the number of top items is $k = \tau n$ for a $\tau$-quantile, while for a different set of size $n' \neq n$, this would correspond to $k' = \tau n' \neq k$.

The implementation of the Precision@$k$ algorithm in [15] indirectly acknowledges the problem that the notion of top $k$ does not generalize since the command-line flag requires $k$ to be specified as a fraction of the positive samples. Nevertheless, the formulation of the problem as well as the solution are still in terms of the top $k$ items of the training set. A study of various statistical questions related to the problem of accuracy at the top is discussed by [9]. The authors also present generalization bounds for the specific case of empirical risk minimization (ERM) under some assumptions about the hypothesis set and the distribution. But, to our knowledge, no previous publication has given general learning guarantees for the problem of accuracy in the top quantile scoring items or carefully addressed the corresponding algorithmic problem.

We discuss the formulation of this problem (Section 3.1) and define an algorithm optimizing a convex surrogate of the corresponding loss in the case of linear scoring functions. We discuss the solution of this problem in terms of several simple convex optimization problems and show that these problems can be extended to the case where positive semi-definite kernels are used (Section 3.2). In Section 4, we present a Rademacher complexity analysis of the problem and give margin-based guarantees for our algorithm based on the $\tau$-quantile value of the functions in the hypothesis set. In Section 5, we also report the results of several experiments evaluating the performance of our algorithm. In a comparison in a bipartite setting with several algorithms seeking high precision at the top, our algorithm achieves a better performance in precision at the top. We start with a presentation of notions and notation useful for the discussion in the following sections.

## 2 Preliminaries

Let $\mathcal{X}$ denote the input space and $D$ a distribution over $\mathcal{X} \times \mathcal{X}$. We interpret the presence of a pair $(x, x')$ in the support of $D$ as the preference of $x'$ over $x$. We denote by $S = \big((x_1, x'_1), \ldots, (x_m, x'_m)\big) \in (\mathcal{X} \times \mathcal{X})^m$ a labeled sample of size $m$ drawn i.i.d. according to $D$ and denote by $\widehat{D}$ the corresponding empirical distribution. $D$ induces a marginal distribution over $\mathcal{X}$ that we denote by $D'$, which in the discrete case can be defined via

$$D'(x) = \frac{1}{2} \sum_{x' \in \mathcal{X}} \big( D(x, x') + D(x', x) \big).$$

We also denote by $\widehat{D}'$ the empirical distribution associated to $D'$ based on the sample $S$.

The learning problems we are studying are defined in terms of the top $\tau$-quantile of the values taken by a function $h \colon \mathcal{X} \to \mathbb{R}$, that is a score $q$ such that $\Pr_{x \sim D'}[h(x) > q] = \tau$ (see Figure 1(a)). In general, $q$ is not unique and this equality may hold for all $q$ in an interval $[q_{\min}, q_{\max}]$. We will be particularly interested in the properties of the set of points $x$ whose scores are above a quantile, that is $s_q = \{x \colon h(x) > q\}$. Since for any $(q, q') \in [q_{\min}, q_{\max}]^2$, $s_q$ and $s_{q'}$ differ only by a set of measure zero, the particular choice of $q$ in that interval has no significant consequence. Thus, in what follows, when it is not unique, we will choose the quantile value to be the maximum, $q_{\max}$.

For any $\tau \in [0, 1]$, let $\rho_\tau$ denote the function defined by

$$\forall u \in \mathbb{R}, \quad \rho_\tau(u) = -\tau(u)_- + (1 - \tau)(u)_+,$$

where $(u)_+ = \max(u, 0)$ and $(u)_- = \min(u, 0)$ (see Figure 1(b)). $\rho_\tau$ is convex as a sum of two convex functions since $u \mapsto (u)_+$ is convex, $u \mapsto (u)_-$ concave. We will denote by $\operatorname{argMin}_u f(u)$ the largest minimizer of function $f$. It is known (see for example [17]) that the

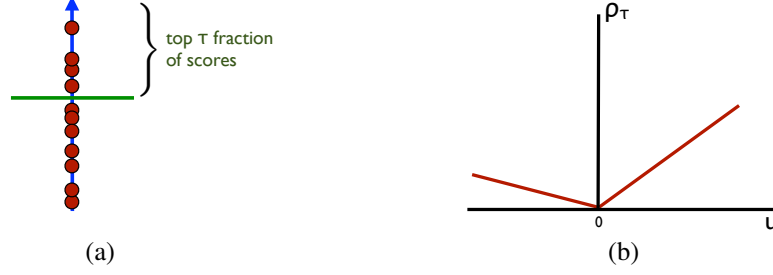

Figure 1: (a) Illustration of the $\tau$-quantile. (b) Graph of function $\rho_\tau$ for $\tau = .25$.

(maximum) $\tau$-quantile value $\widehat{q}$ of a sample of real numbers $X = (u_1, \ldots, u_n) \in \mathbb{R}^n$ can be given by $\widehat{q} = \operatorname{argMin}_{u \in \mathbb{R}} F_\tau(u)$, where $F_\tau$ is the convex function defined for all $u \in \mathbb{R}$ by $F_\tau(u) = \frac{1}{n} \sum_{i=1}^n \rho_\tau(u_i - u)$.

## 3 Accuracy at the top (AATP)

### 3.1 Problem formulation and algorithm

The learning problem we consider is that of *accuracy at the top* (AATP) which consists of achieving an ordering of all items so that items whose scores are among the top $\tau$-quantile are as relevant as possible. Ideally, all preferred items are ranked above the quantile and non-preferred ones ranked below. Thus, the loss or generalization error of a hypothesis $h \colon \mathcal{X} \to \mathbb{R}$ with top $\tau$-quantile value $q_h$ is the average number of non-preferred elements that $h$ ranks above $q_h$ and preferred ones ranked below:

$$R(h) = \frac{1}{2} \operatorname*{E}_{(x,x') \sim D} \left[ 1_{h(x) > q_h} + 1_{h(x') < q_h} \right].$$

$q_h$ can be defined as follows in terms of the distribution $D'$: $q_h = \operatorname{argMin}_{u \in \mathbb{R}} \operatorname{E}_{x \sim D'}[\rho_\tau(h(x) - u)]$. The quantile value $q_h$ depends on the true distribution $D$. To define the empirical error of $h$ for a sample $S = \left( (x_1, x_1'), \ldots, (x_m, x_m') \right) \in (\mathcal{X} \times \mathcal{X})^m$, we will use instead an empirical estimate $\widehat{q}_h$ of $q_h$: $\widehat{q}_h = \operatorname{argMin}_{u \in \mathbb{R}} \operatorname{E}_{x \sim \widehat{D}'}[\rho_\tau(h(x) - u)]$. Thus, we define the empirical error of $h$ for a labeled sample as follows:

$$\widehat{R}(h) = \frac{1}{2m} \sum_{i=1}^m \left[ 1_{h(x_i) > \widehat{q}_h} + 1_{h(x_i') < \widehat{q}_h} \right].$$

We first assume that $\mathcal{X}$ is a subset of $\mathbb{R}^N$ for some $N \geq 1$ and consider a hypothesis set $H$ of linear functions $h \colon \mathbf{x} \mapsto \mathbf{w} \cdot \mathbf{x}$. We will use a surrogate empirical loss taking into consideration how much the score $\mathbf{w} \cdot \mathbf{x}_i$ of a non-preferred item $\mathbf{x}_i$ exceeds $\widehat{q}_h$, and similarly how much lower the score $\mathbf{w} \cdot \mathbf{x}_i'$ for a preferred point $\mathbf{x}_i'$ is than $\widehat{q}_h$, and seek a solution $\mathbf{w}$ minimizing a trade-off of that surrogate loss and the norm squared $\|\mathbf{w}\|^2$. This leads to the following optimization problem for AATP:

$$\min_{\mathbf{w}} \quad \frac{1}{2} \|\mathbf{w}\|^2 + C \left[ \sum_{i=1}^m \left( \mathbf{w} \cdot \mathbf{x}_i - \widehat{q}_{\mathbf{w}} + 1 \right)_+ + \left( \widehat{q}_{\mathbf{w}} - \mathbf{w} \cdot \mathbf{x}_i' + 1 \right)_+ \right] \tag{1}$$

$$\text{subject to} \quad \widehat{q}_{\mathbf{w}} = \operatorname*{argMin}_{u \in \mathbb{R}} Q_\tau(\mathbf{w}, u),$$

where $C \geq 0$ is a regularization parameter and $Q_\tau$ the quantile function defined as follows for a sample $S$, for any $\mathbf{w} \in \mathbb{R}^N$ and $u \in \mathbb{R}$:

$$Q_\tau(\mathbf{w}, u) = \frac{1}{2m} \left[ \sum_{i=1}^m \rho_\tau \left( (\mathbf{w} \cdot \mathbf{x}_i) - u) \right) + \rho_\tau \left( (\mathbf{w} \cdot \mathbf{x}_i') - u) \right) \right].$$

In the following, we will assume that $\tau$ is a multiple of $1/2m$, otherwise it can be rounded to the nearest such value.

### 3.2 Analysis of the optimization problem

Problem (1) is not a convex optimization problem since, while the objective function is convex, the equality constraint is not affine. Here, we further analyze the problem and discuss a solution.

The equality constraint could be written as an infinite number of inequalities of $Q_\tau(\mathbf{w}, \widehat{q}_\mathbf{w}) \leq Q_\tau(\mathbf{w}, u)$ for all $u \in \mathbb{R}$. Observe, however, that the quantile value $q_\mathbf{w}$ must coincide with the score of one of training points $\mathbf{x}_k$ or $\mathbf{x}'_k$, that is $\mathbf{w} \cdot \mathbf{x}_k$ or $\mathbf{w} \cdot \mathbf{x}'_k$. Thus, Problem (1) can be equivalently written with a finite number of constraints as follows:

$$\min_\mathbf{w} \quad \frac{1}{2}\|\mathbf{w}\|^2 + C\Big[\sum_{i=1}^m \big(\mathbf{w} \cdot \mathbf{x}_i - \widehat{q}_\mathbf{w} + 1\big)_+ + \big(\widehat{q}_\mathbf{w} - \mathbf{w} \cdot \mathbf{x}'_i + 1\big)_+\Big]$$

subject to $\quad \widehat{q}_\mathbf{w} \in \{\mathbf{w} \cdot \mathbf{x}_k, \mathbf{w} \cdot \mathbf{x}'_k : k \in [1, m]\}$

$\quad \forall k \in [1, m], Q_\tau(\mathbf{w}, \widehat{q}_\mathbf{w}) \leq Q_\tau(\mathbf{w}, \mathbf{w} \cdot \mathbf{x}_k), \forall k \in [1, m], Q_\tau(\mathbf{w}, \widehat{q}_\mathbf{w}) \leq Q_\tau(\mathbf{w}, \mathbf{w} \cdot \mathbf{x}'_k).$

The inequality constraints do not correspond to non-positivity constraints on convex functions. Thus, the problem is not a standard convex optimization problem, but our analysis leads us to a simple approximate solution for the problem. For convenience, let $(\mathbf{z}_1, \ldots, \mathbf{z}_{2m})$ denote $(\mathbf{x}_1, \ldots, \mathbf{x}_m, \mathbf{x}'_1, \ldots, \mathbf{x}'_m)$. Our method consists of solving the convex quadratic programming (QP) problem for each value of $k \in [1, 2m]$:

$$\min_\mathbf{w} \quad \frac{1}{2}\|\mathbf{w}\|^2 + C\Big[\sum_{i=1}^m \big(\mathbf{w} \cdot \mathbf{x}_i - \widehat{q}_\mathbf{w} + 1\big)_+ + \big(\widehat{q}_\mathbf{w} - \mathbf{w} \cdot \mathbf{x}'_i + 1\big)_+\Big] \qquad (2)$$

subject to $\quad \widehat{q}_\mathbf{w} = \mathbf{w} \cdot \mathbf{z}_k.$

Let $\mathbf{w}_k$ be the solution of Problem (2). For each $k \in [1, 2m]$, we determine the $\tau$-quantile value of the scores $\{\mathbf{w}_k \cdot \mathbf{z}_i : i \in [1, 2m]\}$. This can be checked straightforwardly in time $O(m \log m)$ by sorting the scores. Then, the solution $\mathbf{w}^*$ we return is the $\mathbf{w}_k$ for which $\mathbf{w}_k \cdot \mathbf{z}_k$ is closest to the $\tau$-quantile value, the one for which the objective function is the smallest in the presence of ties. The method for determining $\mathbf{w}^*$ is thus based on the solution of $2m$ simple QPs. Our solution naturally parallelizes so that on a distributed computing environment, the computational time for solving the problem can be reduced to roughly the same as that of solving a single QP.

### 3.3 Kernelized formulation

For any $i \in [1, 2m]$, let $y_i = -1$ if $i \leq m$, $y_i = +1$ otherwise. Then, Problem (2) admits the following equivalent dual optimization problem similar to that of SVMs:

$$\max_\mathbf{\alpha} \quad \sum_{i=1}^{2m} \alpha_i - \frac{1}{2}\sum_{i,j=1}^{2m} \alpha_i \alpha_j y_i y_j (\mathbf{z}_i - \mathbf{z}_k) \cdot (\mathbf{z}_j - \mathbf{z}_k) \qquad (3)$$

subject to: $\forall i \in [1, 2m], \; 0 \leq \alpha_i \leq C,$

which depends only on inner products between points of the training set. The vector $\mathbf{w}$ can be obtained from the solution via $\mathbf{w} = \sum_{i=1}^{2m} \alpha_i y_i (\mathbf{z}_i - \mathbf{z}_k)$. The algorithm can therefore be generalized by using equivalently any positive semi-definite kernel symmetric (PDS) kernel $K : \mathcal{X} \times \mathcal{X} \to \mathbb{R}$ instead of the inner product in the input space, thereby also extending it to the case of non-vectorial input spaces $\mathcal{X}$. The corresponding hypothesis set $H$ is that of linear functions $h : \mathbf{x} \mapsto \mathbf{w} \cdot \mathbf{\Phi}(\mathbf{x})$ where $\mathbf{\Phi} : \mathcal{X} \to \mathbb{H}$ is a feature mapping to a Hilbert space $\mathbb{H}$ associated to $K$ and $\mathbf{w}$ an element of $\mathbb{H}$. In view of (3), for any $k \in [1, 2m]$, the dual problem of (2) can then be expressed as follows:

$$\max_\mathbf{\alpha} \quad \sum_{i=1}^{2m} \alpha_i - \frac{1}{2}\sum_{i,j=1}^{2m} \alpha_i \alpha_j y_i y_j K_k(\mathbf{z}_i, \mathbf{z}_j) \qquad (4)$$

subject to: $\forall i \in [1, 2m], \; 0 \leq \alpha_i \leq C,$

where, for any $k \in [1, 2m]$, $K_k$ is the PDS kernel defined by $K_k : (\mathbf{z}, \mathbf{z}') \mapsto K(\mathbf{z}, \mathbf{z}') - K(\mathbf{z}, \mathbf{z}_k) - K(\mathbf{z}_k, \mathbf{z}') + K(\mathbf{z}_k, \mathbf{z}_k)$. Our solution can therefore also be found in the dual by solving the $2m$ QPs defined by (4).

## 4 Theoretical guarantees

We here present margin-based generalization bounds for the AATP learning problem.

Let $\Phi_\rho\colon \mathbb{R} \to [0;1]$ be the function defined by $\Phi_\rho\colon x \mapsto 1_{x\le 0} + (1 - x/\rho)_+ 1_{x>0}$. For any $\rho > 0$ and $t \in \mathbb{R}$, we define the generalization error $R(h,t)$ and empirical margin loss $\widehat{R}_\rho(h,t)$, both with respect to $t$, by

$$R(h,t) = \frac{1}{2} \mathop{\mathrm{E}}_{(x,x')\sim D} \left[1_{h(x)>t} + 1_{h(x')<t}\right] \quad \widehat{R}_\rho(h,t) = \frac{1}{2m} \sum_{i=1}^m \left[\Phi_\rho(t - h(x_i)) + \Phi_\rho(h(x_i') - t)\right].$$

In particular, $R(h, q_h)$ corresponds to the generalization error and $\widehat{R}_\rho(h, q_h)$ to the empirical margin loss of a hypothesis $h$ for AATP. For any $t > 0$, the empirical margin loss $\widehat{R}_\rho(h,t)$ is upper bounded by the average of the fraction of non-preferred elements $x_i$ that $h$ ranks above $t$ or less than $\rho$ below $t$, and the fraction of preferred ones $x_i'$ it ranks below $t$ or less than $\rho$ above $t$:

$$\widehat{R}_\rho(h,t) \le \frac{1}{2m} \sum_{i=1}^m \left[1_{t-h(x_i)<\rho} + 1_{h(x_i')-t<\rho}\right]. \tag{5}$$

We denote by $D_1$ the marginal distribution of the first element of the pairs in $\mathcal{X} \times \mathcal{X}$ derived from $D$, and by $D_2$ the marginal distribution with respect to the second element. Similarly, $S_1$ is the sample derived from $S$ by keeping only the first element of each pair: $S_1 = (x_1, \ldots, x_m)$ and $S_2$ the one obtained by keeping only the second element: $S_2 = (x_1', \ldots, x_m')$. We also denote by $\mathfrak{R}_m^{D_1}(H)$ the Rademacher complexity of $H$ with respect to the marginal distribution $D_1$, that is $\mathfrak{R}_m^{D_1}(H) = \mathrm{E}[\widehat{\mathfrak{R}}_{S_1}(H)]$, and $\mathfrak{R}_m^{D_2}(H) = \mathrm{E}[\widehat{\mathfrak{R}}_{S_2}(H)]$.

**Theorem 1** *Let $H$ be a set of real-valued functions taking values in $[-M, +M]$ for some $M > 0$. Fix $\tau \in [0,1]$ and $\rho > 0$, then, for any $\delta > 0$, with probability at least $1 - \delta$ over the choice of a sample $S$ of size $m$, each of the following inequalities holds for all $h \in H$ and $t \in [-M, +M]$:*

$$R(h,t) \le \widehat{R}_\rho(h,t) + \frac{1}{\rho} \left(\mathfrak{R}_m^{D_1}(H) + \mathfrak{R}_m^{D_2}(H) + \frac{2M}{\sqrt{m}}\right) + \sqrt{\frac{\log 1/\delta}{2m}}$$

$$R(h,t) \le \widehat{R}_\rho(h,t) + \frac{1}{\rho} \left(\widehat{\mathfrak{R}}_{S_1}(H) + \widehat{\mathfrak{R}}_{S_2}(H) + \frac{2M}{\sqrt{m}}\right) + 3\sqrt{\frac{\log 2/\delta}{2m}}.$$

*Proof.* Let $\widetilde{H}$ be the family of hypotheses mapping $(\mathcal{X} \times \mathcal{X})$ to $\mathbb{R}$ defined by $\widetilde{H} = \{z = (x, x') \mapsto t - h(x)\colon h \in H, t \in [-M, +M]\}$ and similarly $\widetilde{H}' = \{z = (x, x') \mapsto h(x') - t\colon h \in H, t \in [-M, +M]\}$. Consider the two families of functions $\widetilde{\mathcal{H}}$ and $\widetilde{\mathcal{H}}'$ taking values in $[0, 1]$ defined by $\widetilde{\mathcal{H}} = \{\Phi_\rho \circ f\colon f \in \widetilde{H}\}$ and $\widetilde{\mathcal{H}}' = \{\Phi_\rho \circ f\colon f \in \widetilde{H}'\}$. By the general Rademacher complexity bounds for functions taking values in $[0, 1]$ [18, 3, 20], with probability at least $1 - \delta$,

$$\frac{1}{2} \mathrm{E}\left[\Phi_\rho(t - h(x)) + \Phi_\rho(h(x') - t)\right] \le \widehat{R}_\rho(h,t) + 2\mathfrak{R}_m\left(\frac{1}{2}(\widetilde{\mathcal{H}} + \widetilde{\mathcal{H}}')\right) + \sqrt{\frac{\log 1/\delta}{2m}}$$

$$\le \widehat{R}_\rho(h,t) + \mathfrak{R}_m(\widetilde{\mathcal{H}}) + \mathfrak{R}_m(\widetilde{\mathcal{H}}') + \sqrt{\frac{\log 1/\delta}{2m}},$$

for all $h \in H$. Since $1_{u<0} \le \Phi_\rho(u)$ for all $u \in \mathbb{R}$, the generalization error $R(h,t)$ is a lower bound on left-hand side: $R(h,t) \le \frac{1}{2} \mathrm{E}\left[\Phi_\rho(t - h(x)) + \Phi_\rho(h(x') - t)\right]$, we obtain

$$R(h,t) \le \widehat{R}_\rho(h,t) + \mathfrak{R}_m(\widetilde{\mathcal{H}}) + \mathfrak{R}_m(\widetilde{\mathcal{H}}') + \sqrt{\frac{\log 1/\delta}{2m}}.$$

Since $\Phi_\rho$ is $1/\rho$-Lipschitz, by Talagrand's contraction lemma, we have $\mathfrak{R}_m(\widetilde{\mathcal{H}}) \le (1/\rho)\mathfrak{R}_m(\widetilde{H})$ and $\mathfrak{R}_m(\widetilde{\mathcal{H}}') \le (1/\rho)\mathfrak{R}_m(\widetilde{H}')$. By definition of the Rademacher complexity,

$$\mathfrak{R}_m(\widetilde{H}) = \frac{1}{m} \mathop{\mathrm{E}}_{S\sim D^m, \boldsymbol{\sigma}}\left[\sup_{h\in H, t} \sum_{i=1}^m \sigma_i(t - h(x_i))\right] = \frac{1}{m} \mathop{\mathrm{E}}_{S,\boldsymbol{\sigma}}\left[\sup_t \sum_{i=1}^m \sigma_i t + \sup_{h\in H} \sum_{i=1}^m -\sigma_i h(x_i)\right]$$

$$= \frac{1}{m} \mathop{\mathrm{E}}_{\boldsymbol{\sigma}}\left[\sup_{t\in[-M,+M]} t\sum_{i=1}^m \sigma_i\right] + \frac{1}{m} \mathop{\mathrm{E}}_{\boldsymbol{\sigma}}\left[\sup_{h\in H} \sum_{i=1}^m -\sigma_i h(x_i)\right].$$

Since the random variables $\sigma_i$ and $-\sigma_i$ follow the same distribution, the second term coincides with $\mathfrak{R}_m^{D_1}(H)$. The first term can be rewritten and upper bounded as follows using Jensen's inequality:

$$\frac{1}{m}\operatorname*{E}_{\boldsymbol{\sigma}}\left[\sup_{-M\leq t\leq M}\sum_{i=1}^m \sigma_i t\right] = \frac{M}{m}\sum_{\sum_{i=1}^m \sigma_i>0}\Pr[\boldsymbol{\sigma}]\sum_{i=1}^m\sigma_i - \frac{M}{m}\sum_{\sum_{i=1}^m\sigma_i<0}\Pr[\boldsymbol{\sigma}]\sum_{i=1}^m\sigma_i$$

$$= \frac{M}{m}\operatorname*{E}_{\boldsymbol{\sigma}}\left[\left|\sum_{i=1}^m\sigma_i\right|\right] \leq \frac{M}{m}\operatorname*{E}_{\boldsymbol{\sigma}}\left[\left(\sum_{i=1}^m\sigma_i\right)^2\right]^{\frac{1}{2}} = \frac{M}{m}\operatorname*{E}_{\boldsymbol{\sigma}}\left[\sum_{i=1}^m\sigma_i^2\right]^{\frac{1}{2}} = \frac{M}{\sqrt{m}}.$$

Note that, by the Kahane-Khintchine inequality, the last upper bound used is tight modulo a constant $(1/\sqrt{2})$. Similarly, we can show that $\mathfrak{R}_m(\widetilde{H}') \leq \mathfrak{R}_m^{D_2}(H) + M/\sqrt{m}$. This proves the first inequality of the theorem; the second inequality can be derived from the first one using the standard bound relating the empirical and true Rademacher complexity. □

Since the bounds of the theorem hold uniformly for all $t \in [-M, +M]$, they hold in particular for any quantile value $q_h$.

**Corollary 1 (Margin bounds for AATP)** *Let $H$ be a set of real-valued functions taking values in $[-M, +M]$ for some $M > 0$. Fix $\tau \in [0, 1]$ and $\rho > 0$, then, for any $\delta > 0$, with probability at least $1 - \delta$ over the choice of a sample $S$ of size $m$, for all $h \in H$ it holds that:*

$$R(h) \leq \widehat{R}_\rho(h, q_h) + \frac{1}{\rho}\left(\mathfrak{R}_m^{D_1}(H) + \mathfrak{R}_m^{D_2}(H) + \frac{2M}{\sqrt{m}}\right) + \sqrt{\frac{\log 1/\delta}{2m}}$$

$$R(h) \leq \widehat{R}_\rho(h, q_h) + \frac{1}{\rho}\left(\widehat{\mathfrak{R}}_{S_1}(H) + \widehat{\mathfrak{R}}_{S_2}(H) + \frac{2M}{\sqrt{m}}\right) + 3\sqrt{\frac{\log 2/\delta}{2m}}.$$

A more explicit version of this corollary can be derived for kernel-based hypotheses (Appendix A).

In the results of the previous theorem and corollary, the right-hand side of the generalization bounds is expressed in terms of the empirical margin loss with respect to the true quantile value $q_h$, which is upper bounded (see (5)) by half the fraction of non-preferred points in the sample whose score is above $q_h - \rho$ and half the fraction of the preferred points whose score is less than $q_h + \rho$. These fractions are close to the same fractions with $q_h$ replaced with $\widehat{q}_h$ since the probability that a score falls between $q_h$ and $\widehat{q}_h$ can be shown to be uniformly bounded by a term in $O(1/\sqrt{m})$.[1] Altogether, this analysis provides a strong support for our algorithm which is precisely seeking to minimize the sum of an empirical margin loss based on the quantile and a term that depends on the complexity, as in the right-hand side of the learning guarantees above.

## 5 Experiments

This section reports the results of experiments with our AATP algorithm on several datasets. To measure the effectiveness of our algorithm, we compare it to two other algorithms, the INFINITE-PUSH algorithm [1] and the SVMPERF algorithm [15], which are both algorithms seeking to emphasize the accuracy near the top. Our experiments are carried out using three data sets from the UC Irvine Machine Learning Repository http://archive.ics.uci.edu/ml/datasets.html: Ionosphere, Housing, and Spambase. (Results for Spambase can be found in Appendix C). In addition, we use the TREC 2003 (LETOR 2.0) data set which is available for download from the following Microsoft Research URL: http://research.microsoft.com/letor.

All the UC Irvine data sets we experiment with are for two-group classification problems. From these we construct bipartite ranking problems where a preference pair consists of one positive and one negative example. To explicitly indicate the dependency on the quantile, we denote by $q_\tau$ the value of the top $\tau$-th quantile of the score distribution of a hypothesis. We will use $N$ to denote the number of instances in a particular data set, as well as $s_i$, $i = 1, \ldots, N$, to denote the particular score values. If $n_+$ denotes the number of positive examples in the data set and $n_-$ denotes the number of negative examples, then $N = n_+ + n_-$ and the number of preferences is $m = n_+ n_-$.

Table 1: Ionosphere data: for each top quantile $\tau$ and each evaluation metric, the three rows correspond to AATP (top), SVMPERF(middle) and INFINITEPUSH (bottom). For the INFINITEPUSH algorithm we only report mean values over the folds.

| $\tau$ (%) | P@$\tau$ | AP | DCG@$\tau$ | NDCG@$\tau$ | Positives@top |
|---|---|---|---|---|---|
| 19 | $0.89 \pm 0.04$ | $0.86 \pm 0.03$ | $29.21 \pm 0.10$ | $0.92 \pm 0.06$ | $12.1 \pm 12.5$ |
| | $0.89 \pm 0.06$ | $0.83 \pm 0.04$ | $28.88 \pm 1.37$ | $0.89 \pm 0.11$ | $6.00 \pm 11.1$ |
| | $0.85$ | $0.80$ | $27.83$ | $0.85$ | $10.32$ |
| 14 | $0.91 \pm 0.05$ | $0.84 \pm 0.03$ | $28.15 \pm 0.95$ | $0.91 \pm 0.07$ | $13.31 \pm 12.5$ |
| | $0.82 \pm 0.11$ | $0.79 \pm 0.04$ | $27.02 \pm 1.37$ | $0.75 \pm 0.16$ | $4.10 \pm 11.1$ |
| | $0.87$ | $0.80$ | $27.91$ | $0.87$ | $11.51$ |
| 9.50 | $0.93 \pm 0.06$ | $0.84 \pm 0.03$ | $28.15 \pm 0.95$ | $0.91 \pm 0.09$ | $13.31 \pm 12.49$ |
| | $0.77 \pm 0.18$ | $0.79 \pm 0.04$ | $27.02 \pm 1.35$ | $0.70 \pm 0.21$ | $4.50 \pm 10.9$ |
| | $0.90$ | $0.80$ | $27.90$ | $0.89$ | $11.51$ |
| 5 | $0.91 \pm 0.14$ | $0.84 \pm 0.03$ | $28.15 \pm 0.95$ | $0.89 \pm 0.15$ | $13.31 \pm 12.49$ |
| | $0.66 \pm 0.27$ | $0.79 \pm 0.04$ | $27.02 \pm 1.36$ | $0.60 \pm 0.30$ | $4.60 \pm 11.0$ |
| | $0.86$ | $0.81$ | $27.90$ | $0.87$ | $11.59$ |
| 1 | $0.85 \pm 0.24$ | $0.84 \pm 0.03$ | $28.15 \pm 0.95$ | $0.88 \pm 0.19$ | $13.30 \pm 12.53$ |
| | $0.35 \pm 0.41$ | $0.79 \pm 0.04$ | $27.02 \pm 1.36$ | $0.34 \pm 0.41$ | $4.50 \pm 11.0$ |
| | $0.85$ | $0.80$ | $27.91$ | $0.86$ | $11.50$ |

## 5.1 Implementation

We solved the convex optimization problems (2) using the CVX solver http://cvxr.com/. As already noted, the AATP problem can be solved efficiently using a distributed computing environment. The convex optimization problem of the INFINITEPUSH algorithm (see (3.9) of [1]) can also be solved using CVX. However, this optimization problem has as many variables as the product of the numbers of positively and negatively labeled instances ($n_+ n_-$), which makes it prohibitive to solve for large data sets within a runtime of a few days. Thus, we experimented with the INFINITEPUSH algorithm only on the Ionosphere data set. Finally, for SVM-PERF's training and score prediction we used the binary executables downloaded from the URL http://www.cs.cornell.edu/people/tj and used the SVMPERF's settings that are the closest to our optimization formulation. Thus, we used L1-norm for slack variables and allowed the constraint cache and the tolerance for termination criterion to grow in order to control the algorithm's convergence, especially for larger values of the regularization constant.

## 5.2 Evaluation measures

To evaluate and compare the AATP, INFINITEPUSH, and SVMPERF algorithms, we used a number of standard metrics: Precision at the top (P@$\tau$), Average Precision (AP), Number of positives at the absolute top (Positives@top), Discounted Cumulative Gain (DCG@$\tau$), and Normalized Discounted Cumulative Gain (NDCG@$\tau$). Definitions are included in Appendix B.

## 5.3 Ionosphere data

The data set's 351 instances represent radar signals collected from phased antennas, where 'good' signals (225 positively labeled instances) are those that reflect back toward the antennas and 'bad' signals (126 negatively labeled instances) are those that pass through the ionosphere. The data has 34 features. We split the data set into 10 independent sets of instances, say $S_1, \ldots, S_{10}$. Then, we ran 10 experiments, where we used 3 consecutive sets for learning and the rest (7 sets) for testing. We evaluated and compared the algorithms for 5 different top quantiles $\tau \in \{19, 14, 9.5, 5, 1\}$ (%), which would correspond to the top $20, 15, 10, 5, 1$ items, respectively. For each $\tau$, the regularization parameter $C$ was selected based on the average value of P@$\tau$. The performance of AATP is significantly better than that of the other algorithms, particularly for the smallest top quantiles. The two main criteria on which to evaluate the AATP algorithm are *Precision at the top*, (P@$\tau$), and *Number of positive at the top*, (Positives@top). For $\tau = 5\%$ the AATP algorithm obtains a stellar 91% accuracy with an average of 13.3 positive elements at the top (Table 1).

Table 2: Housing data: for each quantile value $\tau$ and each evaluation metric, there are two rows corresponding to AATP (top) and SVMPERF(bottom).

| $\tau$ (%) | P@$\tau$ | AP | DCG@$\tau$ | NDCG@$\tau$ | Positives@top |
|---|---|---|---|---|---|
| 6 | $0.14 \pm 0.05$ | $0.11 \pm 0.03$ | $4.64 \pm 0.40$ | $0.13 \pm 0.08$ | $0.20 \pm 0.45$ |
|   | $0.13 \pm 0.05$ | $0.10 \pm 0.02$ | $4.81 \pm 0.46$ | $0.16 \pm 0.09$ | $0.21 \pm 0.45$ |
| 5 | $0.17 \pm 0.07$ | $0.10 \pm 0.03$ | $4.69 \pm 0.26$ | $0.16 \pm 0.07$ | $0.00 \pm 0.00$ |
|   | $0.12 \pm 0.10$ | $0.09 \pm 0.03$ | $4.76 \pm 0.60$ | $0.16 \pm 0.14$ | $0.20 \pm 0.48$ |
| 4 | $0.19 \pm 0.13$ | $0.12 \pm 0.03$ | $4.83 \pm 0.45$ | $0.18 \pm 0.15$ | $0.00 \pm 0.00$ |
|   | $0.14 \pm 0.05$ | $0.10 \pm 0.02$ | $4.66 \pm 0.25$ | $0.13 \pm 0.07$ | $0.00 \pm 0.00$ |
| 3 | $0.20 \pm 0.12$ | $0.10 \pm 0.03$ | $4.70 \pm 0.26$ | $0.18 \pm 0.11$ | $0.00 \pm 0.00$ |
|   | $0.17 \pm 0.12$ | $0.09 \pm 0.02$ | $4.65 \pm 0.40$ | $0.18 \pm 0.13$ | $0.00 \pm 0.00$ |
| 2 | $0.23 \pm 0.10$ | $0.10 \pm 0.03$ | $4.69 \pm 0.26$ | $0.19 \pm 0.11$ | $0.00 \pm 0.00$ |
|   | $0.25 \pm 0.17$ | $0.10 \pm 0.03$ | $4.89 \pm 0.48$ | $0.27 \pm 0.16$ | $0.20 \pm 0.46$ |
| 1 | $0.20 \pm 0.27$ | $0.12 \pm 0.03$ | $4.80 \pm 0.45$ | $0.17 \pm 0.23$ | $0.00 \pm 0.00$ |
|   | $0.30 \pm 0.27$ | $0.09 \pm 0.02$ | $4.74 \pm 0.56$ | $0.29 \pm 0.27$ | $0.20 \pm 0.45$ |

## 5.4 Housing data

The Boston Housing data set has 506 examples, 35 positive and 471 negative, described by 13 features. We used feature 4 as the binary target value. Two thirds of the data instances was randomly selected and used for training, and the rest for testing. We created 10 experimental folds analogously as in the case of the Ionosphere data. The Housing data is very unbalanced with less than 7% positive examples. For this dataset we obtain results very comparable to SVMPERF for the very top quantiles, see Table 2. Naturally, the standard deviations are large as a result of the low percentage of positive examples, so the results are not always significant. For higher top quantiles, e.g., top 4%, the AATP algorithm significantly outperforms SVMPERF, obtaining 19% accuracy at the top (P@$\tau$). For the highest top quantiles the difference in performance between the two algorithms is not significant.

## 5.5 LETOR 2.0

This data set corresponds to a relatively hard ranking problem, with an average of only 1% relevant query-URL pairs per query. It consists of 5 folds. Our Matlab implementation (with CVX) of the algorithms prevented us from trying our approach on larger data sets. Hence from each training fold we randomly selected 500 items for training. For testing, we selected 1000 items at random from the test fold. Here, we only report results for P@1%. SVMPERF obtained an accuracy of $1.5\% \pm 1.5\%$ while the AATP algorithm obtained an accuracy of $4.6\% \pm 2.4\%$. This significantly better result indicates the power of the algorithm proposed.

# 6 Conclusion

We presented a series of results for the problem of accuracy at the top quantile, including an AATP algorithm, a margin-based theoretical analysis in support of that algorithm, and a series of experiments with several data sets demonstrating the effectiveness of our algorithm. These results are of practical interest in applications where the accuracy among the top quantile is sought. The analysis of problems based on other loss functions depending on the top $\tau$-quantile scores is also likely to benefit form the theoretical and algorithmic results we presented.

The optimization algorithm we discussed is highly parallelizable, since it is based on solving $2m$ independent QPs. Our initial experiments reported here were carried out using Matlab with CVX, which prevented us from evaluating our approach on larger data sets, such as the full LETOR 2.0 data set. However, we have now designed a solution for very large $m$ based on the ADMM (Alternating Direction Method of Multipliers) framework [4]. We have implemented that solution and will present and discuss it in future work.

## Footnotes

[1]Note that the Bahadur-Kiefer representation is known to provide a uniform convergence bound on the difference of the true and empirical quantiles when the distribution admits a density [2, 16], a stronger result than what is needed in our context.

# References

[1] S. Agarwal. The infinite push: A new support vector ranking algorithm that directly optimizes accuracy at the absolute top of the list. In *Proceedings of the SIAM International Conference on Data Mining*, 2011.

[2] R. R. Bahadur. A note on quantiles in large samples. *Annals of Mathematical Statistics*, 37, 1966.

[3] P. L. Bartlett and S. Mendelson. Rademacher and Gaussian complexities: Risk bounds and structural results. *Journal of Machine Learning Research*, 3:2002, 2002.

[4] S. Boyd, N. Parikh, E. Chu, B. Peleato, and J. Eckstein. Distributed optimization and statistical learning via the alternating direction method of multipliers. *Foundations and Trends in Machine Learning*, 3(1):1–122, 2011.

[5] S. Boyd and L. Vandenberghe. *Convex Optimization*. Cambridge University Press, 2004.

[6] J. S. Breese, D. Heckerman, and C. M. Kadie. Empirical analysis of predictive algorithms for collaborative filtering. In *UAI '98: Proceedings of the Fourteenth Conference on Uncertainty in Artificial Intelligence*. Morgan Kaufmann, 1998.

[7] C. Burges, T. Shaked, E. Renshaw, A. Lazier, M. Deeds, N. Hamilton, and G. Hullender. Learning to rank using gradient descent. In *Proceedings of the 22nd international conference on Machine learning*, ICML '05, pages 89–96, New York, NY, USA, 2005. ACM.

[8] C. J. C. Burges, R. Ragno, and Q. V. Le. Learning to rank with nonsmooth cost functions. In *NIPS*, pages 193–200, 2006.

[9] S. Clémençon and N. Vayatis. Ranking the best instances. *Journal of Machine Learning Research*, 8:2671–2699, 2007.

[10] D. Cossock and T. Zhang. Statistical analysis of Bayes optimal subset ranking. *IEEE Transactions on Information Theory*, 54(11):5140–5154, 2008.

[11] K. Crammer and Y. Singer. PRanking with ranking. In *Neural Information Processing Systems (NIPS 2001)*. MIT Press, 2001.

[12] Y. Freund, R. Iyer, R. E. Schapire, and Y. Singer. An efficient boosting algorithm for combining preferences. *J. Mach. Learn. Res.*, 4, December 2003.

[13] R. Herbrich, K. Obermayer, and T. Graepel. *Advances in Large Margin Classifiers*, chapter Large Margin Rank Boundaries for Ordinal Regression. MIT Press, 2000.

[14] T. Joachims. Optimizing search engines using clickthrough data. In *Proceedings of the eighth ACM SIGKDD international conference on Knowledge discovery and data mining*, KDD '02, pages 133–142, New York, NY, USA, 2002. ACM.

[15] T. Joachims. A support vector method for multivariate performance measures. In *ICML*, pages 377–384, 2005.

[16] J. Kiefer. On Bahadur's representation of sample quantiles. *Annals of Mathematical Statistics*, 38, 1967.

[17] R. Koenker. *Quantile Regression*. Cambridge University Press, 2005.

[18] V. Koltchinskii and D. Panchenko. Empirical margin distributions and bounding the generalization error of combined classifiers. *Annals of Statistics*, 30, 2002.

[19] Q. V. Le, A. Smola, O. Chapelle, and C. H. Teo. Optimization of ranking measures. Unpublished, 2009.

[20] M. Mohri, A. Rostamizadeh, and A. Talwalkar. *Foundations of Machine Learning*. The MIT Press, 2012.

[21] C. Rudin, C. Cortes, M. Mohri, and R. E. Schapire. Margin-based ranking meets boosting in the middle. In *COLT*, pages 63–78, 2005.

[22] I. Tsochantaridis, T. Joachims, T. Hofmann, and Y. Altun. Large margin methods for structured and interdependent output variables. *Journal of Machine Learning Research*, 6:1453–1484, 2005.

